# A probabilistic model of auditory space representation in the barn owl

**Brian J. Fischer**
Dept. of Electrical and Systems Eng.
Washington University in St. Louis
St. Louis, MO 63110
fischerb@pcg.wustl.edu

**Charles H. Anderson**
Department of Anatomy and Neurbiology
Washington University in St. Louis
St. Louis, MO 63110
cha@pcg.wustl.edu

## Abstract

The barn owl is a nocturnal hunter, capable of capturing prey using auditory information alone [1]. The neural basis for this localization behavior is the existence of auditory neurons with spatial receptive fields [2]. We provide a mathematical description of the operations performed on auditory input signals by the barn owl that facilitate the creation of a representation of auditory space. To develop our model, we first formulate the sound localization problem solved by the barn owl as a statistical estimation problem. The implementation of the solution is constrained by the known neurobiology.

## 1   Introduction

The barn owl shows great accuracy in localizing sound sources using only auditory information [1]. The neural basis for this localization behavior is the existence of auditory neurons with spatial receptive fields called space specific neurons [2]. Experimental evidence supports the hypothesis that spatial selectivity in auditory neurons arises from tuning to a specific combination of the interaural time difference (ITD) and the interaural level difference (ILD) [3]. Still lacking, however, is a complete account of how ITD and ILD spectra are integrated across frequency to give rise to spatial selectivity. We describe a computational model of the operations performed on the auditory input signals leading to an initial representation of auditory space. We develop the model in the context of a statistical estimation formulation of the localization problem that the barn owl must solve. We use principles of signal processing and estimation theory to guide the construction of the model, but force the implementation to respect neurobiological constraints.

## 2   The environment

The environment consists of $N_s$ point sources and a source of ambient noise. Each point source is defined by a sound signal, $s_i(t)$, and a direction $(\theta_i, \phi_i)$ where $\theta_i$ is the azimuth and $\phi_i$ is the elevation of the source relative to the owl's head. In general, source location may change over time. For simplicity, however, we assume that source locations are fixed. Source signals can be broadband or narrowband. Signals with onsets are modeled as

broadband noise signals modulated by a temporal envelope, $s_i(t) = [\sum_{n=1}^{N_i} w_{in}(t)]n_i(t)$, where $w_{in}(t) = A_{in}e^{-\frac{1}{2}(t-c_{in})^2/\sigma_{in}^2}$ and $n_i(t)$ is Gaussian white noise bandlimited to 12 kHz (see figure $(4A)$). The ambient noise is described below.

## 3  Virtual Auditory Space

The first step in the localization process is the location-dependent mapping of source signals to the received pressure waveforms at the eardrums. For a given source location, the system describing the transformation of a source signal to the waveform received at the eardrum is well approximated by a linear system. This system is characterized by its transfer function called the head related transfer function (HRTF) or, equivalently, by its impulse response, the head related impulse response (HRIR). Additionally, when multiple sources are present the composite waveform at each ear is the sum of the waveforms received due to each source alone. Therefore, we model the received pressure waveforms at the ears as

$$r_L(t) = \sum_{i=1}^{N_s} h_{L(\theta_i,\phi_i)}(t) * s_i(t) + n_L(t) \text{ and } r_R(t) = \sum_{i=1}^{N_s} h_{R(\theta_i,\phi_i)}(t) * s_i(t) + n_R(t) \quad (1)$$

where $h_{L(\theta,\phi)}(t)$ and $h_{R(\theta,\phi)}(t)$ are the HRIRs for the left and right ears, respectively, when the source location is $(\theta, \phi)$, [4], and $n_L(t)$, $n_R(t)$ are the ambient noises experienced by the left and right ears, respectively. For our simulations, the ambient noise for each ear is created using a sample of a natural sound recording of a stream, $s_b(t)$ [5]. The sample is filtered by HRIRs for all locations in the frontal hemisphere, $\Omega$, then averaged so that $n_L(t) = \frac{1}{|\Omega|} \sum_{i\in\Omega} h_{L(\theta_i,\phi_i)}(t) * s_b(t)$ and $n_R(t) = \frac{1}{|\Omega|} \sum_{i\in\Omega} h_{R(\theta_i,\phi_i)}(t) * s_b(t)$.

## 4  Cue Extraction

In our model, location information is not inferred directly from the received signals but is obtained from stimulus-independent binaural location cues extracted from the input signals [6],[7]. The operations used in our model to process the auditory input signals and extract cues are motivated by the known processing in the barn owl's auditory system and by the desire to extract stimulus-independent location cues from the auditory signals that can be used to infer the locations of sound sources.

### 4.1  Cochlear processing

In the first stage of our model, input signals are filtered with a bank of linear band-pass filters. Following linear filtering, input signals undergo half-wave rectification. So, the input signals to the two ears $r_L(t)$ and $r_R(t)$ are decomposed into a set of scalar valued functions $u_L(t, \omega_k)$ and $u_R(t, \omega_k)$ defined by

$$u_L(t, \omega_k) = [f_{\omega_k} \star r_L(t)]_+ \text{ and } u_R(t, \omega_k) = [f_{\omega_k} \star r_R(t)]_+ \quad (2)$$

where $f_{\omega_k}(t)$ is the linear bandpass filter for the channel with center frequency $\omega_k$. Here we use the standard gammatone filter $f_{\omega_k}(t) = t^{\gamma-1}e^{-t/\tau_k}\cos(\omega_k t)$ with $\gamma = 4$ [8]. Following rectification there is a gain control step that is a modified version of the divisive normalization model of Schwartz and Simoncelli [9]. We introduce intermediate variables $\gamma_L(t, \omega_k)$ and $\gamma_R(t, \omega_k)$ that dynamically compute the intensity of the signals within each frequency channel as

$$\dot{\gamma}_L(t, \omega_k) = -\frac{\gamma_L(t, \omega_k)}{\tau} + \frac{u_L(t, \omega_k)}{\sum_n a_{kn}\gamma(t, \omega_n) + \sigma} \quad (3)$$

and

$$\dot{\gamma}_R(t, \omega_k) = -\frac{\gamma_R(t, \omega_k)}{\tau} + \frac{u_R(t, \omega_k)}{\sum_n a_{kn}\gamma(t, \omega_n) + \sigma} \tag{4}$$

where $\gamma(t, \omega_n) = \gamma_L(t, \omega_k) + \gamma_R(t, \omega_k)$. We define the output of the cochlear filter in frequency channel $k$ to be

$$v_L(t, \omega_k) = \frac{u_L(t, \omega_k)}{\sum_n a_{kn}\gamma(t, \omega_n) + \sigma} \text{ and } v_R(t, \omega_k) = \frac{u_R(t, \omega_k)}{\sum_n a_{kn}\gamma(t, \omega_n) + \sigma} \tag{5}$$

for the left and right, respectively. Note that the rectified outputs from the left and right ears, $u_L(t, \omega_k)$ and $u_R(t, \omega_k)$, are normalized by the same term so that binaural disparities are not introduced by the gain control operation. Initial cue extraction operations are performed within distinct frequency channels established by this filtering process.

## 4.2 Level difference cues

The level difference pathway has two stages. First, the outputs of the filter banks are integrated over time to obtain windowed intensity measures for the components of the left and right ear signals. Next, signals from the left and right ears are combined within each frequency channel to measure the location dependent level difference. We compute the intensity of the signal in each frequency channel over a small time window, $w(t)$, as:

$$y_L(t, \omega_k) = \int_0^t v_L(\sigma, \omega_k)w(t - \sigma)d\sigma \text{ and } y_R(t, \omega_k) = \int_0^t v_R(\sigma, \omega_k)w(t - \sigma)d\sigma. \tag{6}$$

We use a simple exponential window $w(t) = e^{-t/\tau}H(t)$ where $H(t)$ is the unit step function.

The magnitude of $y_L(t, \omega_k)$ and $y_R(t, \omega_k)$ vary with both the signal intensity and the gain of the HRIR in the frequency band centered at $\omega_k$. To compute the level difference between the input signals that is introduced by the HRIRs in a manner that is invariant to changes in the intensity of the source signal we compute

$$z(t, \omega_k) = \log(\frac{y_R(t, \omega_k)}{y_L(t, \omega_k)}). \tag{7}$$

## 4.3 Temporal difference cues

We use a modified version of the standard windowed cross correlation operation to measure time differences. Our modifications incorporate three features that model processing in the barn owl's auditory system. First, signals are passed through a saturating nonlinearity to model the saturation of the nucleus magnocellularis (NM) inputs to the nucleus laminaris (NL) [10]. We define $\chi_L(t, \omega_k) = F(v_L(t, \omega_k))$ and $\chi_R(t, \omega_k) = F(v_R(t, \omega_k))$, where $F(\cdot)$ is a saturating nonlinearity. Let $x(t, \omega_k, m)$ denote the value of the cross correlation in frequency channel $k$ at delay index $m \in \{0, \ldots, N\}$, defined by

$$\dot{x}(t, \omega_k, m) = -\frac{x(t, \omega_k, m)}{\tau(y(t, \omega_k))} + [\chi_L(t - \Delta m, \omega_k) + \alpha][\chi_R(t - \Delta(N - m), \omega_k) + \beta]. \tag{8}$$

Here, $\tau(y(t, \omega_k))$ is a time constant that varies with the intensity of the stimulus in the frequency channel where $y(t, \omega_k) = y_L(t, \omega_k) + y_R(t, \omega_k)$. The time constant decreases as $y(t, \omega_k)$ increases, so that for more intense sounds information is integrated over a smaller time window. This operation functions as a gain control and models the inhibition of NL neurons by superior olive neurons [11]. The constants $\alpha, \beta > 0$ are included to reflect the fact that NL neurons respond to monaural stimulation, [12], and are chosen so that at input levels above threshold $(0 - 5 \text{ dB SPL})$ the cross correlation term dominates. We choose the delay increment $\Delta$ to satisfy $\Delta N = 200\mu s$ so that the full range of possible delays is covered.

## 5   Representing auditory space

The general localization problem that the barn owl must solve is that of localizing multiple objects in its environment using both auditory and visual cues. An abstract discussion of a possible solution to the localization problem will motivate our model of the owl's initial representation of auditory space. Let $N_s(t)$ denote the number of sources at time $t$. Assume that each source is characterized by the direction pair $(\theta_i, \phi_i)$ that obeys a dynamical system $(\dot{\theta}_i, \dot{\phi}_i) = f(\theta_i, \phi_i, \mu_i)$ where $\mu_i$ is a noise term and $f : R^3 \to R^2$ is a possibly nonlinear mapping. We assume that $(\theta_i(t), \phi_i(t))$ defines a stationary stochastic process with known density $p(\theta_i, \phi_i)$ [6],[7]. At time $t$, let $\xi_t^a$ denote a vector of cues computed from auditory input and let $\xi_t^v$ denote a vector of cues computed from visual input. The problem is to estimate, at each time, the number and locations of sources in the environment using past measurements of the auditory and visual cues at a finite set of sample times. A simple Bayesian approach is to introduce a minimal state vector $\alpha_t = [\theta(t)\ \phi(t)]^T$ where $\dot{\alpha}_t = f(\alpha_t, \mu_t)$ and compute the posterior density of $\alpha_t$ given the cue measurements. Here the number and locations of sources can be inferred from the existence and placement of multiple modes in the posterior. If we assume that the state sequence $\{\alpha_{t_n}\}$ is a Markov process and that the state is conditionally independent of past cue measurements given the present cue measurement, then we can recursively compute the posterior through a process of prediction and correction described by the equations

$$p(\alpha_{t_n}|\xi_{t_1:t_{n-1}}) = \int\int p(\alpha_{t_n}|\alpha_{t_{n-1}})p(\alpha_{t_{n-1}}|\xi_{t_1:t_{n-1}})d\alpha_{t_{n-1}} \tag{9}$$

$$p(\alpha_{t_n}|\xi_{t_1:t_n}) \propto p(\xi_{t_n}|\alpha_{t_n})p(\alpha_{t_n}|\xi_{t_1:t_{n-1}}) = p(\xi_{t_n}^a|\alpha_{t_n})p(\xi_{t_n}^v|\alpha_{t_n})p(\alpha_{t_n}|\xi_{t_1:t_{n-1}}) \tag{10}$$

where $\xi_t = [\xi_t^a\ \xi_t^v]^T$. This formulation suggests that at each time auditory space can be represented in terms of the likelihood function $p(\xi_t^a|\theta(t), \phi(t))$.

## 6   Combining temporal and intensity difference signals

To facilitate the calculation of the likelihood function over the locations, we introduce compact notation for the cues derived from the auditory signals. Let $\mathbf{x}(t, \omega_k) = [x(t, \omega_k, 0), \ldots, x(t, \omega_k, N)]/\|[x(t, \omega_k, 0), \ldots, x(t, \omega_k, N)]\|$ be the normalized vector of cross correlations computed within frequency channel $k$. Let $\mathbf{x}(t) = [\mathbf{x}(t, \omega_1), \ldots, \mathbf{x}(t, \omega_{N_F})]$ denote the spectrum of cross correlations and let $\mathbf{z}(t) = [z(t, \omega_1), \ldots, z(t, \omega_{N_F})]$ denote the spectrum of level differences where $N_F$ is the number of frequency channels. Let $\xi_t^a = [\mathbf{x}(t)\ \mathbf{z}(t)]^T$. We assume that $\xi_t^a = [\mathbf{x}(t)\ \mathbf{z}(t)]^T = [\bar{\mathbf{x}}(\theta, \phi)\ \bar{\mathbf{z}}(\theta, \phi)]^T + \eta(t)$ where $\bar{\mathbf{x}}(\theta, \phi)$ and $\bar{\mathbf{z}}(\theta, \phi)$ are the expected values of the cross correlation and level difference spectra, respectively, for a single source located at $(\theta, \phi)$, and $\eta(t)$ is Gaussian white noise [6],[7].
Experimental evidence about the nature of auditory space maps in the barn owl suggests that spatial selectivity occurs after both the combination of temporal and level difference cues and the combination of information across frequency [3],[13]. The computational model specifies that the transformation from cues computed from the auditory input signals to a representation of space occurs by performing inference on the cues through the likelihood function

$$p(\xi_t^a|\theta, \phi) = p(\mathbf{x}(t), \mathbf{z}(t)|\theta, \phi) \propto \exp(-\frac{1}{2}\|(\mathbf{x}(t), \mathbf{z}(t)) - (\bar{\mathbf{x}}(\theta, \phi), \bar{\mathbf{z}}(\theta, \phi))\|_{\Sigma_n^{-1}}^2). \tag{11}$$

The known physiology of the barn owl places constraints on how this likelihood function can be computed. First, the spatial tuning of auditory neurons in the optic tectum is consistent with a model where spatial selectivity arises from tuning to combinations of time difference and level difference cues within each frequency channel [14]. This suggests

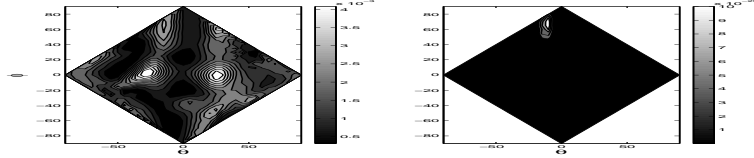

Figure 1: Non-normalized likelihood functions at $t = 26$ ms with sources located at $(-25^o, 0^o)$ and $(0^o, 25^o)$. Source signals are $s_1(t) = A \sum_i \cos(\omega_{i1}(t))$ and $s_2(t) = A \sum_j \cos(\omega_{j2}(t))$ where $\omega_{i1} \neq \omega_{j2}$ for any $i, j$. Left: Linear model of frequency combination. Right: Multiplicative model of frequency combination.

that time and intensity information is initially combined multiplicatively within frequency channels.

Given this constraint we propose two models of the frequency combination step. In the first model of frequency integration we assume that the likelihood is a product of kernels

$$p(\mathbf{x}(t), \mathbf{z}(t)|\theta, \phi) \propto \prod_k K(\mathbf{x}(t, \omega_k), z(t, \omega_k); \theta, \phi). \tag{12}$$

Each kernel is a product of a temporal difference function and a level difference function to respect the first constraint,

$$K(\mathbf{x}(t, \omega_k), z(t, \omega_k); \theta, \phi) = K_x(\mathbf{x}(t, \omega_k); \theta, \phi) K_z(z(t, \omega_k); \theta, \phi). \tag{13}$$

If we require that each kernel is normalized,
$\int \int K(\mathbf{x}(t_*, \omega_k), z(t_*, \omega_k); \theta, \phi) d\mathbf{x}(t_*, \omega_k) dz(t_*, \omega_k) = 1$, for each $t_*$ then the multiplicative model is a factorization of the likelihood into a product of the conditional probabilities $p(\mathbf{x}(t_*, \omega_k), z(t_*, \omega_k)|\theta, \phi)$. The second model is a linear model of frequency integration where the likelihood is approximated by a kernel estimate of the form

$$p(\mathbf{x}(t), \mathbf{z}(t)|\theta, \phi) \propto \sum_k c_k(y(t, \omega_k)) K(\mathbf{x}(t, \omega_k), z(t, \omega_k); \theta, \phi) \tag{14}$$

where each kernel is of the above product form. We again assume that the kernels are normalized, but we weight each kernel by the intensity of the signal in that frequency channel.

Experiments performed in multiple source environments by Takahashi et al. suggest that information is not multiplied across frequency channels [15]. Takahashi et al. measured the response of space specific neurons in the external nucleus of the inferior colliculus under conditions of two sound sources located on the horizontal plane with each signal consisting of a unique combination of sinusoids. Their results suggest that a bump of activity will be present at each source location in the space map. Using identical stimuli (see Table 1 columns A and C in [15]) we compute the likelihood function using the linear model and the multiplicative model. The results shown in figure (1) demonstrate that with a linear model the likelihood function will display a peak corresponding to each source location, but with the multiplicative model only a spurious location that is consistent among the kernels remains and information about the two sources is lost. Therefore, we use a model in which time difference and level difference information is first combined multiplicatively within frequency channels and is then summed across frequency.

# 7 Examples

## 7.1 Parameters

In each example stimuli are presented for $100$ ms and HRIRs for owl $884$ recorded by Keller et al., [4], are used to generate the input signals. We use six gammatone filters for each ear

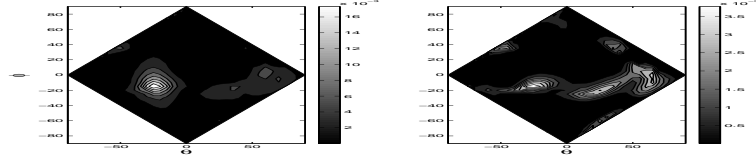

Figure 2: Non-normalized likelihood functions at $t = 21.1$ ms for a single source located at $(-25^o, -15^o)$. Left: Broadband source signal at 50 dB SPL. Right: Source signal is a 7 kHz tone at 50 dB SPL.

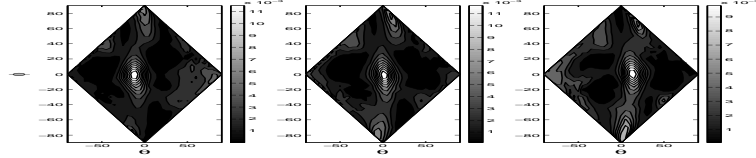

Figure 3: Non-normalized likelihood functions under conditions of summing localization. In each case sources are located at $(-20^o, 0^o)$ and $(20^o, 0^o)$ and produce scaled versions of the same waveform. Left: Left signal at 50 dB SPL, right signal at 40 dB SPL. Center: Left signal at 50 dB SPL, right signal at 50 dB SPL. Right: Left signal at 40 dB SPL, right signal at 50 dB SPL.

with center frequencies $\{4.22, 5.14, 6.16, 7.26, 8.47, 9.76\}$ kHz, and $Q_{10}$ values chosen to match the auditory nerve fiber data of Köppl [16]. In each example we use a Gaussian form for the temporal and level difference kernels, $K_x(\mathbf{x}(t, \omega_k); \theta, \phi) \propto \exp(-\frac{1}{2}\|\mathbf{x}(t, \omega_k) - \bar{\mathbf{x}}(\theta, \phi)\|^2/\sigma^2)$ and $K_z(z(t, \omega_k); \theta, \phi) \propto \exp(-\frac{1}{2}\|z(t, \omega_k) - \bar{z}(\theta, \phi)\|^2/\sigma^2)$ where $\sigma^2 = 0.1$. The terms $\bar{\mathbf{x}}(\theta, \phi)$ and $\bar{z}(\theta, \phi)$ correspond to the time average of the cross correlation and level difference cues for a broadband noise stimulus. Double polar coordinates are used to describe source locations. Only locations in the frontal hemisphere are considered. Ambient noise is present at 10 dB SPL.

## 7.2 Single source

In figure (2) we show the approximate likelihood function of equation (19) at a single time during the presentation of a broadband noise stimulus and a 7 kHz tone from direction $(-25^o, -15^o)$. In response to the broadband signal there is a peak at the source location. In response to the tone there is a peak at the true location and significant peaks near $(60^o, -5^o)$ and $(20^o, -25^o)$.

## 7.3 Multiple sources

In figure (3) we show the response of our model under the condition of summing localization. The top signal shown in figure $(4A)$ was presented from $(-20^o, 0^o)$ and $(20^o, 0^o)$ with no delay between the two sources, but with varied intensities for each signal. In each case there is a single phantom bump at an intermediate location that is biased toward the more intense source.

In figure (4) we simulate an echoic environment where the signal at the top of $4A$ is presented from $(-20^o, 0^o)$ and a copy delayed by 2 ms shown at the bottom of $4A$ is presented from $(20^o, 0^o)$. We plot the likelihood function at the three times indicated by vertical dotted lines in $4A$. At the first time the initial signal dominates and there is a peak at the location of the leading source. At the second time when both the leading and lagging sounds have similar envelope amplitudes there is a phantom bump at an intermediate, al-

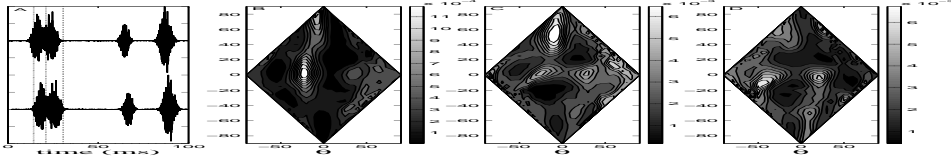

Figure 4: Non-normalized likelihoods under simulated echoic conditions. The leading signal is presented from $(-20^o, 0^o)$ and the lagging source from $(20^o, 0^o)$. Both signals are presented at 50 dB SPL. A: The top signal is the leading signal and the bottom is the lagging. Vertical lines show times at which the likelihood function is plotted in B,C,D. B: Likelihood at $t = 14.3$ ms. C: Likelihood at $t = 21.1$ ms. D: Likelihood at $t = 30.6$ ms.

though elevated, location. At the third time where the lagging source dominates there are peaks at both the leading and lagging locations.

## 8  Discussion

We used a Bayesian approach to the localization problem faced by the barn owl to guide our modeling of the computational operations supporting sound localization in the barn owl. In the context of our computational model, auditory space is initially represented in terms of a likelihood function parameterized by time difference and level difference cues computed from the auditory input signals.

In transforming auditory cues to spatial locations, the model relies on stimulus invariance in the cue values achieved by normalizing the cross correlation vector and computing a ratio of the left and right signal intensities within each frequency channel. It is not clear from existing experimental data where or if this invariance occurs in the barn owl's auditory system.

In constructing a model of the barn owl's solution to the estimation problem, the operations that we employ are constrained to be consistent with the known physiology. As stated above, physiological data is consistent with the multiplication of temporal difference and level difference cues in each frequency channel, but not with multiplication across frequency. This model does not explain, however, across frequency nonlinearities that occur in the processing of temporal difference cues [17].

The likelihood function used in our model is a linear approximation to the likelihood specified in equation (11). The multiplicative model clearly does not explain the response of the space map to multiple sound sources producing spectrally nonoverlapping signals [15]. The linear approximation may reflect the requirement to function in a multiple source environment. We must more precisely define the multi-target tracking problem that the barn owl solves and include all relevant implementation constraints before interpreting the nature of the approximation.

The tuning of space specific neurons to combinations of ITD and ILD has been interpreted as a multiplication of ITD and ILD related signals [3]. Our model suggests that, to be consistent with known physiology, the multiplication of ITD and ILD signals occurs in the medial portion of the lateral shell of the central nucleus of the inferior colliculus before frequency convergence [13]. Further experiments must be done to determine if the multiplication is a network property of the first stage of lateral shell neurons or if multiplication occurs at the level of single neurons in the lateral shell.

We simulated the model's responses under conditions of summing localization and simulated echoes. The model performs as expected for two simultaneous sources with a phantom bump occurring in the likelihood function at a location intermediate between the two source locations. Under simulated echoic conditions the likelihood shows evidence for both the leading and lagging source, but only the leading source location appears alone.

This suggests that with this instantaneous estimation procedure the lagging source would be perceptible as a source location, however, possibly less so than the leading. It is likely that a feedback mechanism, such as the Bayesian filtering described in equations (14) and (15), will need to be included to explain the decreased perception of lagging sources.

## Acknowledgments

We thank Kip Keller, Klaus Hartung, and Terry Takahashi for providing the head related transfer functions. We thank Mike Lewicki for providing the natural sound recordings. This work was supported by the Mathers Foundation.

## References

[1] Payne, R.S., "Acoustic location of prey by barn owls (Tyto alba).", J. Exp. Biol., 54: 535-573, 1971.

[2] Knudsen, E.I., Konishi, M., "A neural map of auditory space in the owl.", Science, 200: 795-797, 1978.

[3] Peña, J.L., Konishi, M., "Auditory receptive fields created by multiplication.", Science, 292: 249-252, 2001.

[4] Keller, C.H., Hartung, K., Takahashi, T.T., "Head-related transfer functions of the barn owl: measurement and neural responses.", Hearing Research, 118: 13-34, 1998.

[5] Lewicki, M.S., "Efficient coding of natural sounds.", Nature Neurosci., 5(4): 356-363, 2002.

[6] Martin, K.D., "A computational model of spatial hearing.", Masters thesis, MIT, 1995.

[7] Duda, R.O., "Elevation dependence of the interaural transfer function.", In Gilkey, R. and Anderson, T.R. (eds.), Binaural and Spatial Hearing, 49-75, 1994.

[8] Slaney, M., "Auditory Toolbox.", Apple technical report 45, Apple Computer Inc., 1994.

[9] Schwartz, O., Simoncelli, E.P., "Natural signal statistics and sensory gain control.", Nature Neurosci., 4(8): 819-825, 2001.

[10] Sullivan, W.E., Konishi, M., "Segregation of stimulus phase and intensity coding in the cochlear nucleus of the barn owl.", J. Neurosci., 4(7): 1787-1799, 1984.

[11] Yang, L., Monsivais, P., Rubel, E.W., "The superior olivary nucleus and its influence on nucleus laminaris: A source of inhibitory feedback for coincidence detection in the avian auditory brainstem.", J. Neurosci., 19(6): 2313-2325, 1999.

[12] Carr, C.E., Konishi, M., "A circuit for detection of interaural time differences in the brain stem of the barn owl." J. Neurosci., 10(10): 3227-3246, 1990.

[13] Mazer, J.A., "Integration of parallel processing streams in the inferior colliculus of the barn owl.", Ph.D thesis, Caltech 1995.

[14] Brainard, M.S., Knudsen, E.I., Esterly, S.D., "Neural derivation of sound source location: Resolution of spatial ambiguities in binaural cues.", J. Acoust. Soc. Am., 91(2): 1015-1026, 1992.

[15] Takahashi, T.T., Keller, C.H., "Representation of multiple sources in the owl's auditory space map.", J. Neurosci., 14(8): 4780-4793, 1994.

[16] Köppl, C., "Frequency tuning and spontaneous activity in the auditory nerve and cochlear nucleus magnocellularis of the barn owl Tyto alba.", J. Neurophys., 77: 364-377, 1997.

[17] Takahashi, T.T., Konishi, M., "Selectivity for interaural time difference in the owl's midbrain.", J. Neurosci., 6(12): 3413-3422, 1986.
